# COMBINED NEURAL NETWORKS FOR TIME SERIES ANALYSIS

**Iris Ginzburg** and **David Horn**
School of Physics and Astronomy
Raymond and Beverly Sackler Faculty of Exact Science
Tel-Aviv University
Tel-Aviv 96678, Israel

## Abstract

We propose a method for improving the performance of any network designed to predict the next value of a time series. We advocate analyzing the deviations of the network's predictions from the data in the training set. This can be carried out by a secondary network trained on the time series of these residuals. The combined system of the two networks is viewed as the new predictor. We demonstrate the simplicity and success of this method, by applying it to the sunspots data. The small corrections of the secondary network can be regarded as resulting from a Taylor expansion of a complex network which includes the combined system. We find that the complex network is more difficult to train and performs worse than the two-step procedure of the combined system.

## 1 INTRODUCTION

The use of neural networks for computational tasks is based on the idea that the efficient way in which the nervous system handles memory and cognition is worth immitating. Artificial implementations are often based on a single network of mathematical neurons. We note, however, that in biological systems one can find collections of consecutive networks, performing a complicated task in several stages, with later stages refining the performance of earlier ones. Here we propose to follow this strategy in artificial applications.

We study the analysis of time series, where the problem is to predict the next element on the basis of previous elements of the series. One looks then for a functional relation

$$y_n = f(y_{n-1}, y_{n-2}, \cdots, y_{n-m}).$$ (1)

This type of representation is particularly useful for the study of dynamical systems. These are characterized by a common continuous variable, time, and many correlated degrees of freedom which combine into a set of differential equations. Nonetheless, each variable can in principle be described by a lag-space representation of the type 1 . This is valid even if the $y = y(t)$ solution is unpredictable as in chaotic phenomena.

Weigend Huberman and Rumelhart (1990) have studied the experimental series of yearly averages of sunspots activity using this approach. They have realized the lag-space representation on an $(m, d, 1)$ network, where the notation implies a hidden layer of $d$ sigmoidal neurons and one linear output. Using $m = 12$ and a weight-elimination method which led to $d = 3$, they obtained results which compare favorably with the leading statistical model (Tong and Lim, 1980). Both models do well in predicting the next element of the sunspots series. Recently, Nowlan and Hinton (1992) have shown that a significantly better network can be obtained if the training procedure includes a complexity penalty term in which the distribution of weights is modelled as a mixture of multiple gaussians whose parameters vary in an adaptive manner as the system is being trained.

We propose an alternative method which is capable of improving the performance of neural networks: train another network to predict the errors of the first one, to uncover and remove systematic correlations that may be found in the solution given by the trained network, thus correcting the original predictions. This is in agreement with the general philosophy mentioned at the beginning, where we take from Nature the idea that the task does not have to be performed by one complicated network; it is advantageous to break it into stages of consecutive analysis steps. Starting with a network which is trained on the sunspots data with back-propagation, we show that the processed results improve considerably and we find solutions which match the performance of Weigend et. al.

## 2   CONSTRUCTION OF THE PRIMARY NETWORK

Let us start with a simple application of back-propagation to the construction of a neural network describing the sunspots data which are normalized to lie between 0 and 1. The network is assumed to have one hidden layer of sigmoidal neurons, $h_i\ i = 1, \cdots, d$, which receives the input of the nth vector:

$$h_i = \sigma(\sum_{j=1}^{m} w_{ij} y_{n-j} - \theta_i)$$ (2)

The output of the network, $p_n$, is constructed linearly,

$$p_n = \sum_{i=1}^{d} w_i h_i - \theta.$$ (3)

The error-function which we minimize is defined by

$$E = \frac{1}{2} \sum_{n=m+1}^{N} (p_n - y_n)^2 \tag{4}$$

where we try to equate $p_n$, the prediction or output of the network, with $y_n$, the nth value of the series. This is the appropriate formulation for a training set of $N$ data points which are viewed as $N - m$ strings of length $m$ used to predict the point following each string.

We will work with two sets of data points. One will be labelled $T$ and be used for training the network, and the other $P$ will be used for testing its predictive power. Let us define the average error by

$$\epsilon_S = \sqrt{\frac{1}{\|S\|} \sum_{n \in S} (p_n - y_n)^2} \tag{5}$$

where the set $S$ is either $T$ or $P$. An alternative parameter was used by Weigend et. al. , in which the error is normalized by the standard deviation of the data. This leads to an *average relative variance (arv)* which is related to the average error through

$$arv_S = \frac{\epsilon_S^2}{\sigma_S^2}. \tag{6}$$

Following Weigend et. al. we choose $m = 12$ neurons in the first layer and $\|T\| = 220$ data points for the training set. The following $\|P\| = 35$ years are used for testing the predictions of our network. We use three sigmoidal units in the hidden layer and run with a slow convergence rate for 7000 periods. This is roughly where cross-validation would indicate that a minimum is reached. The starting parameters of our networks are chosen randomly. Five examples of such networks are presented in Table 1.

## 3   THE SECONDARY NETWORK

Given the networks constructed above, we investigate their deviations from the desired values

$$q_n = y_n - p_n. \tag{7}$$

A standard statistical test for the quality of any predictor is the analysis of the correlations between consecutive errors. If such correlations are found, the predictor must be improved. The correlations reflect a systematic deviation of the primary network from the true solution. We propose not to improve the primary network by modifying its architecture but to add to it a secondary network which uses the residuals $q_n$ as its new data. The latter is being trained only after the training session of the primary network has been completed.

Clearly one may expect some general relation of the type

$$q_n = f(q_{n-1}, q_{n-2}, \cdots, q_{n-l}; y_{n-1}, y_{n-2}, \cdots, y_{n-l}) \tag{8}$$

to exist. Looking for a structure of this kind enlarges considerably the original space in which we searched for a solution to 1 . We wish the secondary network

to do a modest task, therefore we assume that much can be gained by looking at the interdependence of the residuals $q_n$ on themselves. This reduces the problem to finding the best values of

$$r_n = f_1(q_{n-1}, q_{n-2}, \cdots, q_{n-l}) \tag{9}$$

which would minimize the new error function

$$E_2 = \frac{1}{2} \sum_{n=l+1}^{N} (r_n - q_n)^2. \tag{10}$$

Alternatively, one may try to express the residual in terms of the functional values

$$r_n = f_2(y_{n-1}, y_{n-2}, \cdots, y_{n-l}) \tag{11}$$

minimizing again the expression 10 .

When the secondary network completes its training, we propose to view

$$t_n = p_n + r_n \tag{12}$$

as the new prediction of the combined system. We will demonstrate that a major improvement can be obtained already with a linear perceptron. This means that the linear regression

$$r_n = \sum_{i=1}^{l} \alpha_i^1 q_{n-i} + \beta^1 \tag{13}$$

or

$$r_n = \sum_{i=1}^{l} \alpha_i^2 y_{n-i} + \beta^2 \tag{14}$$

is sufficient to account for a large fraction of the systematic deviations of the primary networks from the true function that they were trained to represent.

## 4  NUMERICAL RESULTS

We present in Table 1 five examples of results of (12,5,1) networks, i.e. $m = 12$ inputs, a hidden layer of three sigmoidal neurons and a linear output neuron. These five examples were chosen from 100 runs of simple back-propagation networks with random initial conditions by selecting the networks with the smallest $R$ values (Ginzburg and Horn, 1992). This is a weak constraint which is based on letting the network generate a large sequence of data by iterating its own predictions, and selecting the networks whose distribution of function values is the closest to the corresponding distribution of the training set.

The errors of the primary networks, in particular those of the prediction set $\epsilon_P$, are quite higher than those quoted by Weigend et. al. who started out from a (12,8,1) network and brought it down through a weight elimination technique to a (12,5,1) structure. They have obtained the values $\epsilon_T = 0.059$ $\epsilon_P = 0.06$. We can reduce our errors and reach the same range by activating a secondary network with $l = 11$ to perform the linear regression (3.6) on the residuals of the predictions of the primary network. The results are the primed errors quoted in the table. Characteristically we observe a reduction of $\epsilon_T$ by $3 - 4\%$ and a reduction of $\epsilon_P$ by more than 10%.

| # | $\epsilon_T$ | $\epsilon'_T$ | $\epsilon_P$ | $\epsilon'_P$ |
|---|---|---|---|---|
| 1 | 0.0614 | 0.0587 | 0.0716 | 0.0620 |
| 2 | 0.0600 | 0.0585 | 0.0721 | 0.0663 |
| 3 | 0.0611 | 0.0580 | 0.0715 | 0.0621 |
| 4 | 0.0621 | 0.0594 | 0.0698 | 0.0614 |
| 5 | 0.0616 | 0.0589 | 0.0681 | 0.0604 |

### Table 1

Error parameters of five networks. The unprimed errors are those of the primary networks. The primed errors correspond to the combined system which includes correction of the residuals by a linear perceptron with $l = 11$, which is an autoregressions of the residuals. Slightly better results for the short term predictions are achieved by corrections based on regression of the residuals on the original input vectors, when the regression length is 13 (Table 2).

| # | $\epsilon_T$ | $\epsilon'_T$ | $\epsilon_P$ | $\epsilon'_P$ |
|---|---|---|---|---|
| 1 | 0.061 | 0.059 | 0.072 | 0.062 |
| 2 | 0.060 | 0.059 | 0.072 | 0.065 |
| 3 | 0.061 | 0.058 | 0.072 | 0.062 |
| 4 | 0.062 | 0.060 | 0.070 | 0.061 |
| 5 | 0.062 | 0.059 | 0.068 | 0.059 |

### Table 2

Error parameters for the same five networks. The primed errors correspond to the combined system which includes correction of the residuals by a linear perceptron based on original input vectors with $l = 13$.

## 5    LONG TERM PREDICTIONS

When short term prediction is performed, the output of the original network is corrected by the error predicted by the secondary network. This can be easily generalized to perform long term predictions by feeding the corrected output produced by the combined system of both networks back as input to the primary network. The corrected residuals predicted by the secondary network are viewed as the residuals needed as further inputs if the secondary network is the one performing autoregression of residuals. We run both systems based on regression on residuals and regression on functional values to produce long term predictions.

In table 3 we present the results of this procedure for the case of a secondary network performing regression on residuals. The errors of the long term predictions are averaged over the test set $P$ of the next 35 years. We see that the errors of the primary networks are reduced by about 20%. The quality of these long term predictions is within the range of results presented by Weigend et. al. Using the regression on (predicted) functional values, as in Eq. 14, the results are improved by up to 15% as shown in Table 4.

| # | $\epsilon_2$ | $\epsilon_2'$ | $\epsilon_5$ | $\epsilon_5'$ | $\epsilon_{11}$ | $\epsilon_{11}'$ |
|---|------|------|------|------|------|------|
| 1 | 0.118 | 0.098 | 0.162 | 0.109 | 0.150 | 0.116 |
| 2 | 0.118 | 0.106 | 0.164 | 0.125 | 0.131 | 0.101 |
| 3 | 0.117 | 0.099 | 0.164 | 0.112 | 0.136 | 0.099 |
| 4 | 0.116 | 0.099 | 0.152 | 0.107 | 0.146 | 0.120 |
| 5 | 0.113 | 0.097 | 0.159 | 0.112 | 0.147 | 0.123 |

## Table 3

Long term predictions into the future. $\epsilon_n$ denotes the average error of $n$ time steps predictions over the $P$ set. The unprimed errors are those of the primary networks. The primed errors correspond to the combined system which includes correction of the residuals by a linear perceptron.

| # | $\epsilon_2$ | $\epsilon_2'$ | $\epsilon_5$ | $\epsilon_5'$ | $\epsilon_{11}$ | $\epsilon_{11}'$ |
|---|------|------|------|------|------|------|
| 1 | 0.118 | 0.098 | 0.162 | 0.107 | 0.150 | 0.101 |
| 2 | 0.118 | 0.104 | 0.164 | 0.117 | 0.131 | 0.089 |
| 3 | 0.117 | 0.098 | 0.164 | 0.108 | 0.136 | 0.086 |
| 4 | 0.117 | 0.098 | 0.152 | 0.105 | 0.146 | 0.105 |
| 5 | 0.113 | 0.096 | 0.159 | 0.110 | 0.147 | 0.109 |

## Table 4

Long term predictions into the future. The primed errors correspond to the combined system which includes correction of the residuals by a linear perceptron based on the original inputs.

## 6   THE COMPLEX NETWORK

Since the corrections of the secondary network are much smaller than the characteristic weights of the primary network, the corrections can be regarded as resulting from a Taylor expansion of a complex network which includes the combined system. This can be simply implemented in the case of Eq. 14 which can be incorporated in the complex network as direct linear connections from the input layer to the output neuron, in addition to the non-linear hidden layer, i.e.,

$$ t_n = \sum_{i=1}^{d} w_i h_i + \sum_{j=1}^{m} v_j y_{n-j} - \theta \ . \tag{15} $$

We train such a complex network on the same problem to see how it compares with the two-step approach of the combined networks described in the previous chapters.

The results depend strongly on the training rates of the direct connections, as compared with the training rates of the primary connections (i.e. those of the primary network). When the direct connections are trained faster than the primary ones, the result is a network that resembles a linear perceptron, with non-linear

corrections. In this case, the assumption of the direct connections being small corrections to the primary ones no longer holds. The training error and prediction capability of such a network are worse than those of the primary network. On the other hand, when the primary connections are trained using a faster training rate, we expect the final network to be similar in nature to the combined system. Still, the quality of training and prediction of these solutions is not as good as the quality of the combined system, unless a big effort is made to find the correct rates. Typical results of the various systems are presented in Table 5.

| type of network | $\epsilon_t$ | $\epsilon_p$ |
|---|---|---|
| primary network | 0.061 | 0.072 |
| learning rate of linear weights $= 0.1$ | 0.062 | 0.095 |
| learning rate of linear weights $= 0.02$ | 0.061 | 0.068 |
| combined system | 0.058 | 0.062 |

**Table 5**
Short term predictions of various networks. The learning rate of primary weights is 0.04.

The performance of the complex network can be better than that of the primary network by itself, but it is surpassed by the achievements of the combined system.

# 7  DISCUSSION

It is well known that increasing the complexity of a network is not the guaranteed solution to better performance (Geman et. al. 1992). In this paper we propose an alternative which increases very little the number of free parameters, and focuses on the residual errors one wants to eliminate. Still one may raise the question whether this cannot be achieved in one complex network. It can, provided we are allowed to use different updating rates for different connections. In the extreme limit in which one rate supersedes by far the other one, this is equivalent to a disjoint architecture of a combined two-step system. This emphasizes the point that a solution of a feedforward network to any given task depends on the architecture of the network as well as on its training procedure.

The secondary network which we have used was linear, hence it defined a simple regression of the residual on a series of residuals or a series of function values. In both cases the minimum which the network looks for is unique. In the case in which the residual is expressed as a regression on function values, the problem can be recast in a complex architecture. However, the combined procedure guarantees that the linear weights will be small, i.e. we look for a small linear correction to the prediction of the primary network. If one trains all weights of the complex network at the same rate this condition is not met, hence the worse results.

We advocate therefore the use of the two-step procedure of the combined set of networks. We note that the secondary networks perform well on all possible tests: they reduce the training errors, they

improve short term predictions and they do better on long term predictions as well. Since this approach is quite general and can be applied to any time-series forecasting problem, we believe it should be always tried as a correction procedure.

## REFERENCES

Geman, S., Bienenstock, E., & Doursat, R., 1992.   Neural networks and the bias/variance dilemma. Neural Comp. 4, 1–58.

Ginzburg, I. & Horn, D. 1992. Learning the rule of a time series. Int. Journal of Neural Systems 3, 167–177.

Nowlan, S. J. & Hinton, G. E. 1992. Simplifying neural networks by soft weight-sharing. Neural Comp. 4, 473–493.

Tong, H., & Lim, K. S., 1980. Threshold autoregression, limit cycles and cyclical data. J. R. Stat. Soc. B 42, 245.

Weigend, A. S., Huberman, B. A. & Rumelhart, D. E., 1990. Predicting the Future: A Connectionist Approach, Int. Journal of Neural Systems 1, 193–209.
